# Following Curved Regularized Optimization Solution Paths

**Saharon Rosset**
IBM T.J. Watson Research Center
Yorktown Heights, NY 10598
srosset@us.ibm.com

## Abstract

Regularization plays a central role in the analysis of modern data, where non-regularized fitting is likely to lead to over-fitted models, useless for both prediction and interpretation. We consider the design of incremental algorithms which follow paths of regularized solutions, as the regularization varies. These approaches often result in methods which are both efficient and highly flexible. We suggest a general path-following algorithm based on second-order approximations, prove that under mild conditions it remains "very close" to the path of optimal solutions and illustrate it with examples.

## 1  Introduction

Given a data sample $(x_i, y_i)_{i=1}^n$ (with $x_i \in \mathbb{R}^p$ and $y_i \in \mathbb{R}$ for regression, $y_i \in \{\pm 1\}$ for classification), the generic regularized optimization problem calls for fitting models to the data while controlling complexity by solving a penalized fitting problem:

$$(1) \qquad \hat{\beta}(\lambda) \quad = \quad \arg\min_{\beta} \sum_i C(y_i, \beta' x_i) + \lambda J(\beta)$$

where $C$ is a convex loss function and $J$ is a convex model complexity penalty (typically taken to be the $l_q$ norm of $\beta$, with $q \geq 1$).[1]

Many commonly used supervised learning methods can be cast in this form, including regularized 1-norm and 2-norm support vector machines [13, 4], regularized linear and logistic regression (i.e. Ridge regression, lasso and their logistic equivalents) and more. In [8] we show that boosting can also be described as *approximate* regularized optimization, with an $l_1$-norm penalty.

Detailed discussion of the considerations in selecting penalty and loss functions for regularized fitting is outside the scope of this paper. In general, there are two main areas we need to consider in this selection:

1. *Statistical considerations:* robustness (which affects selection of loss), sparsity ($l_1$-norm penalty encourages sparse solutions) and identifiability are among the questions we should

keep in mind when selecting our formulation.

2. *Computational considerations:* we should be able to solve the problems we pose with the computational resources at our disposal. Kernel methods and boosting are examples of computational tricks that allow us to solve very high dimensional problems – exactly or approximately – with a relatively small cost. In this paper we suggest a new computational approach.

Once we have settled on a loss and penalty, we are still faced with the problem of selecting a "good" regularization parameter $\lambda$, in terms of prediction performance. A common approach is to solve (1) for several values of $\lambda$, then use holdout data (or theoretical approaches, like AIC or SRM) to select a good value. However, if we view the regularized optimization problem as a family of problems, parameterized by the regularization parameter $\lambda$, it allows us to define the "path" of optimal solutions $\{\hat{\beta}(\lambda) : 0 \leq \lambda \leq \infty\}$, which is a 1-dimensional curve through $\mathbb{R}^p$. Path following methods attempt to utilize the mathematical properties of this curve to devise efficient procedures for "following" it and generating the full set of regularized solutions with a (relatively) small computational cost.

As it turns out, there is a family of well known and interesting regularized problems for which efficient *exact* path following algorithms can be devised. These include the lasso [3], 1- and 2-norm support vector machines [13, 4] and many others [9]. The main property of these problems which makes them amenable to such methods is the *piecewise linearity* of the regularized solution path in $\mathbb{R}^p$. See [9] for detailed exposition of these properties and the resulting algorithms.

However, the path following idea can stretch beyond these exact piecewise linear algorithms. The "first order" approach is to use gradient-based approaches. In [8] we have described boosting as an approximate gradient-based algorithm for following $l_1$-norm regularized solution paths. [6] suggest a gradient descent algorithm for finding an optimal solution for a fixed value of $\lambda$ and are seemingly unaware that the path they are going through is of independent interest as it consists of approximate (alas very approximate) solutions to $l_1$-regularized problems. Gradient-based methods, however, can only follow regularized paths under strict and non-testable conditions, and theoretical "closeness" results to the optimal path are extremely difficult to prove for them (see [8] for details).

In this paper, we suggest a general second-order algorithm for following "curved" regularized solution paths (i.e. ones that cannot be followed exactly by piecewise-linear algorithms). It consists of iteratively changing the regularization parameter, while making a single Newton step at every iteration towards the optimal penalized solution, for the current value of $\lambda$. We prove that if both the loss and penalty are "nice" (in terms of bounds on their relevant derivatives in the relevant region), then the algorithm is guaranteed to stay "very close" to the true optimal path, where "very close" is defined as:

> If the change in the regularization parameter at every iteration is $\epsilon$, then the solution path we generate is guaranteed to be within $O(\epsilon^2)$ from the true path of penalized optimal solutions

In section 2 we present the algorithm, and we then illustrate it on $l_1$- and $l_2$-regularized logistic regression in section 3. Section 4 is devoted to a formal statement and proof outline of our main result. We discuss possible extensions and future work in section 5.

## 2 Path following algorithm

We assume throughout that the loss function $C$ is twice differentiable. Assume for now also that the penalty $J$ is twice differentiable (this assumption does not apply to the $l_1$-norm penalty which is of great interest and we address this point later). The key to our

method are the normal equations for (1):

$$\nabla C(\hat{\beta}(\lambda)) + \lambda \nabla J(\hat{\beta}(\lambda)) = 0 \tag{2}$$

Our algorithm iteratively constructs an approximate solution $\beta_t^{(\epsilon)}$ by taking "small" Newton-Raphson steps trying to maintain (2) as the regularization changes. Our main result in this paper is to show, both empirically and theoretically, that for small $\epsilon$, the difference $\|\beta_t^{(\epsilon)} - \hat{\beta}(\lambda_0 + \epsilon \cdot t)\|$ is small, and thus that our method successfully tracks the path of optimal solutions to (1).

Algorithm 1 gives a formal description of our quadratic tracking method. We start from a solution to (1) for some fixed $\lambda_0$ (e.g. $\hat{\beta}(0)$, the non-regularized solution). At each iteration we increase $\lambda$ by $\epsilon$ and take a single Newton-Raphson step towards the solution to (2) with the new $\lambda$ value in step 2(b).

**Algorithm 1** *Approximate incremental quadratic algorithm for regularized optimization*

1. *Set $\beta_0^{(\epsilon)} = \hat{\beta}(\lambda_0)$, set $t = 0$.*

2. *While ($\lambda_t < \lambda_{max}$)*

   (a) $\lambda_{t+1} = \lambda_t + \epsilon$

   (b) $\beta_{t+1}^{(\epsilon)} =$
   $$\beta_t^{(\epsilon)} - \left[ \nabla^2 C(\beta_t^{(\epsilon)}) + \lambda_{t+1} \nabla^2 J(\beta_t^{(\epsilon)}) \right]^{-1} \left[ \nabla C(\beta_t^{(\epsilon)}) + \lambda_{t+1} \nabla J(\beta_t^{(\epsilon)}) \right]$$

   (c) $t = t + 1$

## 2.1 The $l_1$-norm penalty

The $l_1$-norm penalty, $J(\beta) = \|\beta\|_1$, is of special interest because of its favorable statistical properties (e.g. [2]) and its widespread use in popular methods, such as the lasso [10] and 1-norm SVM [13]. However it is not differentiable and so our algorithm does not apply to $l_1$-penalized problems directly.

To understand how we can generalize Algorithm 1 to this situation, we need to consider the Karush-Kuhn-Tucker (KKT) conditions for optimality of the optimization problem implied by (1). It is easy to verify that the normal equations (2) can be replaced by the following KKT-based condition for $l_1$-norm penalty:

$$|\nabla C(\hat{\beta}(\lambda))_j| < \lambda \ \Rightarrow \ \hat{\beta}(\lambda)_j = 0 \tag{3}$$
$$\hat{\beta}(\lambda)_j \neq 0 \ \Rightarrow \ |\nabla C(\hat{\beta}(\lambda))_j| = \lambda \tag{4}$$

these conditions hold for any differentiable loss and tell us that at each point on the path we have a set $\mathcal{A}$ of non-0 coefficients which corresponds to the variables whose current "generalized correlation" $|\nabla C(\hat{\beta}(\lambda))_j|$ is maximal and equal to $\lambda$. All variables with smaller generalized correlation have 0 coefficient at the optimal penalized solution for this $\lambda$. Note that the $l_1$-norm penalty is twice differentiable everywhere except at 0. So if we carefully manage the set of non-0 coefficients according to these KKT conditions, we can still apply our algorithm in the lower-dimensional subspace spanned by non-0 coefficients only.

Thus we get Algorithm 2, which employs the Newton approach of Algorithm 1 for twice differentiable penalty, limited to the sub-space of "active" coefficients denoted by $\mathcal{A}$. It adds to Algorithm 1 updates for the "add variable to active set" and "remove variable from

active set" events, when a variable becomes "highly correlated" as defined in (4) and when a coefficient hits 0 , respectively. [2]

**Algorithm 2** *Approximate incremental quadratic algorithm for regularized optimization with lasso penalty*

1. *Set $\beta_0^{(\epsilon)} = \hat{\beta}(\lambda_0)$, set $t = 0$, set $\mathcal{A} = \{j : \hat{\beta}(\lambda_0)_j \neq 0\}$.*

2. *While ($\lambda_t < \lambda_{max}$)*

   (a) *$\lambda_{t+1} = \lambda_t + \epsilon$*

   (b)

   $$\beta_{t+1}^{(\epsilon)} = \beta_t^{(\epsilon)} - \left[\nabla^2 C(\beta_t^{(\epsilon)})_{\mathcal{A}}\right]^{-1} \cdot \left[\nabla C(\beta_t^{(\epsilon)})_{\mathcal{A}} + \lambda_{t+1} sgn(\beta_t^{(\epsilon)})_{\mathcal{A}}\right]$$

   (c) *$\mathcal{A} = \mathcal{A} \cup \{j \notin \mathcal{A} : \nabla C(\beta_{t+1}^{(\epsilon)})_j > \lambda_{t+1}\}$*

   (d) *$\mathcal{A} = \mathcal{A} - \{j \in \mathcal{A} : |\beta_{t+1,j}^{(\epsilon)}| < \delta\}$*

   (e) *$t = t + 1$*

## 2.2 Computational considerations

For a fixed $\lambda_0$ and $\lambda_{max}$, Algorithms 1 and 2 take $O(1/\epsilon)$ steps. At each iteration they need to calculate the Hessians of both the loss and the penalty at a typical computational cost of $O(n \cdot p^2)$; invert the resulting $p \times p$ matrix at a cost of $O(p^3)$; and perform the gradient calculation and multiplication, which are $o(n \cdot p^2)$ and so do not affect the complexity calculation. Since we implicitly assume throughout that $n \geq p$, we get overall complexity of $O(n \cdot p^2/\epsilon)$. The choice of $\epsilon$ represents a tradeoff between computational complexity and accuracy (in section 4 we present theoretical results on the relationship between $\epsilon$ and the accuracy of the path approximation we get). In practice, our algorithm is practical for problems with up to several hundred predictors and several thousand observations. See the example in section 3.

It is interesting to compare this calculation to the obvious alternative, which is to solve $O(1/\epsilon)$ regularized problems (1) separately, using a Newton-Raphson approach, resulting in the same complexity (assuming the number of Newton-Raphson iterations for finding each solution is bounded). There are several reasons why our approach is preferable:

- The number of iterations until convergence of Newton-Raphson may be large even if it does converge. Our algorithm guarantees we stay very close to the optimal solution path with a single Newton step at each new value of $\lambda$.

- Empirically we observe that in some cases our algorithm is able to follow the path while direct solution for some values of $\lambda$ fails to converge. We assume this is related to various numeric properties of the specific problems being solved.

- For the interesting case of $l_1$-norm penalty and a "curved" loss function (like logistic log-likelihood), there is no direct Newton-Raphson algorithm. Re-formulating the problem into differentiable form requires doubling the dimensionality. Using our Algorithm 2, we can still utilize the same Newton method, with significant computational savings when many coefficients are 0 and we work in a lower-dimensional subspace.

On the flip side, our results in section 4 below indicate that to guarantee successful tracking we require $\epsilon$ to be small, meaning the number of steps we do in the algorithm may be significantly larger than the number of distinct problems we would typically solve to select $\lambda$ using a non-path approach.

### 2.3 Connection to path following methods from numerical analysis

There is extensive literature on path-following methods for solution paths of general parametric problems. A good survey is given in [1]. In this context, our method can be described as a "predictor-corrector" method with a redundant first order predictor step. That is, the corrector step starts from the previous approximate solution. These methods are recognized as attractive options when the functions defining the path (in our case, the combination of loss and penalty) are "smooth" and "far from linear". These conditions for efficacy of our approach are reflected in the regularity conditions for the closeness result in Section 4.

## 3 Example: $l_2$- and $l_1$-penalized logistic regression

Regularized logistic regression has been successfully used as a classification and probability estimation approach [11, 12]. We first illustrate applying our quadratic method to this regularized problem using a small subset of the "spam" data-set, available from the UCI repository (`http://www.ics.uci.edu/~mlearn/MLRepository.html`) which allows us to present some detailed diagnostics. Next, we apply it to the full "spam" data-set, to demonstrate its time complexity on bigger problems.

We first choose five variables and 300 observations and track the solution paths to two regularized logistic regression problems with the $l_2$-norm and the $l_1$-norm penalties:

$$(5) \qquad \hat{\beta}(\lambda) = \arg\min_{\beta} \log(1 + \exp\{-y_i\beta'x_i\}) + \lambda\|\beta\|_2^2$$

$$(6) \qquad \hat{\beta}(\lambda) = \arg\min_{\beta} \log(1 + \exp\{-y_i\beta'x_i\}) + \lambda\|\beta\|_1$$

Figure 1 shows the solution paths $\beta^{(\epsilon)}(t)$ generated by running Algorithms 1 and 2 on this data using $\epsilon = 0.02$ and starting at $\lambda = 0$, i.e. from the non-regularized logistic regression solution. The interesting graphs for our purpose are the ones on the right. They represent the "optimality gap":

$$\mathbf{e}_t = \frac{\nabla C(\beta_t^{(\epsilon)})}{\nabla J(\beta_t^{(\epsilon)})} + \epsilon \cdot t$$

where the division is done componentwise (and so the five curves in each plot correspond to the five variables we are using). Note that the optimal solution $\hat{\beta}(t\epsilon)$ is uniquely defined by the fact that (2) holds and therefore the "optimality gap" is equal to zero componentwise at $\hat{\beta}(t\epsilon)$. By convexity and regularity of the loss and the penalty, there is a correspondence between small values of $\mathbf{e}$ and small distance $\|\beta^{(\epsilon)}(t) - \hat{\beta}(t\epsilon)\|$. In our example we observe that the components of $\mathbf{e}$ seem to be bounded in a small region around 0 for both paths (note the small scale of the $y$ axis in both plots — the maximal error is under $10^{-3}$). We conclude that on this simple example our method tracks the optimal solution paths well, both for the $l_1$- and $l_2$-regularized problems. The plots on the left show the actual coefficient paths — the curve in $\mathbb{R}^5$ is shown as five coefficient traces in $\mathbb{R}$, each corresponding to one variable, with the non-regularized solution (identical for both problems) on the extreme left.

Next, we run our algorithm on the full "spam" data-set, containing $p = 57$ predictors and $n = 4601$ observations. For both the $l_1$- and $l_2$-penalized paths we used

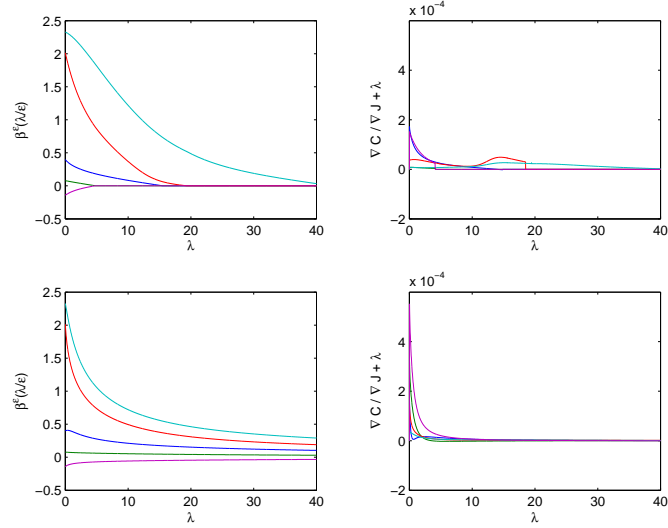

Figure 1: Solution paths (left) and optimality criterion (right) for $l_1$ penalized logistic regression (top) and $l_2$ penalized logistic regression (bottom). These result from running Algorithms 2 and 1, respectively, using $\epsilon = 0.02$ and starting from the non-regularized logistic regression solution (i.e. $\lambda = 0$)

$\lambda_0 = 0, \lambda_{max} = 50, \epsilon = 0.02$, and the whole path was generated in under 5 minutes using a Matlab implementation on an IBM T-30 Laptop. Like in the small scale example, the "optimality criterion" was uniformly small throughout the two paths, with none of its 57 components exceeding $10^{-3}$ at any point.

## 4 Theoretical closeness result

In this section we prove that our algorithm can track the path of true solutions to (1). We show that under regularity conditions on the loss and penalty (which hold for all the candidates we have examined), if we run Algorithm 1 with a specific step size $\epsilon$, then we remain within $O(\epsilon^2)$ of the true path of optimal regularized solutions.

**Theorem 1** *Assume $\lambda_0 > 0$, then for $\epsilon$ small enough and under regularity conditions on the derivatives of $C$ and $J$,*

$$\forall 0 < c < \lambda_{max} - \lambda_0 \ , \ \ \|\beta^{(\epsilon)}(c/\epsilon) - \hat{\beta}(\lambda_0 + c)\| = O(\epsilon^2)$$

*So there is a* uniform *bound $O(\epsilon^2)$ on the error which* does not *depend on c.*

**Proof** We give the details of the proof in Appendix A of [7]. Here we give a brief review of the main steps.

Similar to section 3 we define the "optimality gap":

(7)
$$\left| (\frac{\nabla C(\beta_t^{(\epsilon)})}{\nabla J(\beta_t^{(\epsilon)})})_j + \lambda_t \right| = e_{tj}$$

Also define a "regularity constant" $M$, which depends on $\lambda_0$ and the first, second and third derivatives of the loss and penalty.

The proof is presented as a succession of lemmas:

**Lemma 2** *Let $u_1 = M \cdot p \cdot \epsilon^2$, $u_t = M(u_{t-1} + \sqrt{p} \cdot \epsilon)^2$, then: $\|\mathbf{e}_t\|_2 \leq u_t$*

This lemma gives a recursive expression bounding the error in the optimality gap (7) as the algorithm proceeds. The proof is based on separate Taylor expansions of the numerator and denominator of the ratio $\frac{\nabla C}{\nabla J}$ in the optimality gap and some tedious algebra.

**Lemma 3** *If $\sqrt{p}\epsilon M \leq 1/4$ then $u_t \nearrow \frac{1}{2M} - \sqrt{p} \cdot \epsilon - \frac{\sqrt{1-4\sqrt{p}\cdot\epsilon M}}{2M} = O(\epsilon^2)$ , $\forall t$*

This lemma shows that the recursive bound translates to a uniform $O(\epsilon^2)$ bound, if $\epsilon$ is small enough. The proof consists of analytically finding the fixed point of the increasing series $u_t$.

**Lemma 4** *Under regularity conditions on the penalty and loss functions in the neighborhood of the solutions to (1), the $O(\epsilon^2)$ uniform bound of lemma 3 translates to an $O(\epsilon^2)$ uniform bound on $\|\beta^{(\epsilon)}(c/\epsilon) - \hat{\beta}(\lambda_0 + c)\|$*

Finally, this lemma translates the optimality gap bound to an actual closeness result. This is proven via a Lipschitz argument.

## 4.1 Required regularity conditions

Regularity in the loss and the penalty is required in the definition of the regularity constant $M$ and in the translation of the $O(\epsilon^2)$ bound on the "optimality gap" into one on the distance from the path in lemma 4. The exact derivation of the regularity conditions is highly technical and lengthy. They require us to bound the norm of third derivative "hyper-matrices" for the loss and the penalty as well as the norms of various functions of the gradients and Hessians of both (the boundedness is required only in the neighborhood of the optimal path where our approximate path can venture, obviously). We also need to have $\lambda_0 > 0$ and $\lambda_{max} < \infty$. Refer to Appendix A of [7] for details. Assuming that $\lambda_0 > 0$ and $\lambda_{max} < \infty$ these conditions hold for every interesting example we have encountered, including:

- Ridge regression and the lasso (that is, $l_2$- and $l_1$- regularized squared error loss).
- $l_1$- and $l_2$-penalized logistic regression. Also Poisson regression and other exponential family models.
- $l_1$- and $l_2$-penalized exponential loss.

Note that in our practical examples above we have started from $\lambda_0 = 0$ and our method still worked well. We observe in figure 1 that the tracking algorithm indeed suffers the biggest inaccuracy for the small values of $\lambda$, but manages to "self correct" as $\lambda$ increases.

# 5 Extensions

We have described our method in the context of linear models for supervised learning. There are several natural extensions and enhancements to consider.

**Basis expansions and Kernel methods**

Our approach obviously applies, as is, to models that are linear in basis expansions of the original variables (like wavelets or kernel methods) as long as $p < n$ is preserved. However, the method can easily be applied to high (including infinite) dimensional kernel versions of regularized models where RKHS theory applies. We know that the solution path is fully within the span of the representer functions, that is the columns of the Kernel matrix. With

a kernel matrix $K$ with columns $k_1, ..., k_n$ and the standard $l_2$-norm penalty, the regularized problem becomes:

$$\hat{\alpha}(\lambda) = \arg\min_{\alpha} \sum_i C(y_i, \alpha' k_i) + \lambda \alpha' K \alpha$$

so the penalty now also contains the Kernel matrix, but this poses no complications in using Algorithm 1. The only consideration we need to keep in mind is the computational one, as our complexity is $O(n^3/\epsilon)$. So our method is fully applicable and practical for kernel methods, as long as the number of observations, and the resulting kernel matrix, are not too large (up to several hundreds).

### Unsupervised learning

There is no reason to limit the applicability of this approach to supervised learning. Thus, for example, adaptive density estimation using negative log-likelihood as a loss can be regularized and the solution path be tracked using our algorithm.

### Computational tricks

The computational complexity of our algorithm limits its applicability to large problems. To improve its scalability we primarily need to reduce the effort in the Hessian calculation and inversion. The obvious suggestion here would be to keep the Hessian part of step 2(b) in Algorithm 1 fixed for many iterations and change the gradient part only, then update the Hessian occasionally. The clear disadvantage would be that the "closeness" guarantees would no longer hold. We have not tried this in practice but believe it is worth pursuing.

**Acknowledgements.** The author thanks Noureddine El Karoui for help with the proof and Jerome Friedman, Giles Hooker, Trevor Hastie and Ji Zhu for helpful discussions.

## Footnotes

[1]We assume a linear model in (1), but this is much less limiting than it seems, as the model can be linear in basis expansions of the original predictors, and so our approach covers Kernel methods, wavelets, boosting and more

[2]When a coefficient hits 0 it not only hits a non-differentiability point in the penalty, it also ceases to be maximally correlated as defined in (4). A detailed proof of this fact and the rest of the "accounting" approach can be found in [9]

## References

[1] Allgower, E. L. & Georg, K. (1993). Continuation and path following. *Acta Numer.*, 2:164

[2] Donoho, D., Johnstone, I., Kerkyachairan, G. & Picard, D. (1995). Wavelet shrinkage: Asymptopia? *Annals of Statistics*

[3] Efron, B., Hastie, T., Johnstone, I. & Tibshirani, R.(2004). Least Angle Regression. *Annals of Statistics* .

[4] Hastie, T., Rosset, S., Tibshirani, R. & Zhu, J. (2004). The Entire Regularization Path for the Support Vector Machine. *Journal of Machine Learning Research*, 5(Oct):1391–1415.

[5] Hastie, T., Tibshirani, R. & Friedman, J. (2001). *Elements of Stat. Learning*. Springer-Verlag

[6] Kim, Y & Kim, J. (2004) Gradient LASSO for feature selection. ICML-04, to appear.

[7] Rosset, S. (2003). Topics in Regularization and Boosting. PhD thesis, dept. of Statistics, Stanford University.
http://www-stat.stanford.edu/~saharon/papers/PhDThesis.pdf

[8] Rosset, S., Zhu, J. & Hastie,T. (2003). Boosting as a regularized path to a maximum margin classifier. *Journal of Machine Learning Research*, 5(Aug):941-973.

[9] Rosset, S. & Zhu, J. (2003). Piecewise linear regularized solution paths. Submitted.

[10] Tibshirani, R. (1996). Regression shrinkage and selection via the lasso. *JRSSB*

[11] Wahba, G., Gu, C., Wang, Y. & Chappell, R. (1995) Soft Classification, a.k.a. Risk Estimation, via Penalized Log Likelihood and Smoothing Spline Analysis of Variance. *In D.H. Wolpert, editor, The Mathematics of Generalization*.

[12] Zhu, J. & Hastie, T. (2003). Classification of Gene Microarrays by Penalized Logistic Regression. *Biostatistics, to appear.*

[13] Zhu, J., Hastie, T., Rosset, S. & Tibshirani, R. (2004). 1-norm support vector machines. *Neural Information Processing Systems, 16.*